# Suitable is the Best: Task-Oriented Knowledge Fusion in Vulnerability Detection

**Jingjing Wang**
Institute of Systems Engineering,
Academy of Military Sciences, PLA
jennywange1@163.com

**Minhuan Huang**[*]
Institute of Systems Engineering,
Academy of Military Sciences, PLA
darbean@126.com

**Yuanpin Nie**
Institute of Systems Engineering,
Academy of Military Sciences, PLA
yuanpingnie@nudt.edu.cn

**Xiang Li**
Institute of Systems Engineering,
Academy of Military Sciences, PLA
ideal_work@163.com

**Qianjin Du**
Department of Computer Science
and Technology, Tsinghua University
dqj20@mails.tsinghua.edu.cn

**Wei Kong**
School of Information Science and
Engineering, Zhejiang Sci-Tech University
kong_wei@ieee.org

**Huan Deng**
Institute of Systems Engineering,
Academy of Military Sciences, PLA
denghuan619@163.com

**Xiaohui Kuang**
Institute of Systems Engineering,
Academy of Military Sciences, PLA
xiaohui_kuang@163.com

## Abstract

Deep learning technologies have demonstrated remarkable performance in vulnerability detection. Existing works primarily adopt a uniform and consistent feature learning pattern across the entire target set. While designed for general-purpose detection tasks, they lack sensitivity towards target code comprising multiple functional modules or diverse vulnerability subtypes. In this paper, we present a knowledge fusion-based vulnerability detection method (KF-GVD) that integrates specific vulnerability knowledge into the Graph Neural Network feature learning process. KF-GVD achieves accurate vulnerability detection across different functional modules of the Linux kernel and vulnerability subtypes without compromising general task performance. Extensive experiments demonstrate that KF-GVD outperforms SOTAs on function-level and statement-level vulnerability detection across various target tasks, with an average increase of 40.9% in precision and 26.1% in recall. Notably, KF-GVD discovered 9 undisclosed vulnerabilities when employing on C/C++ open-source projects without ground truth.

## 1 Introduction

According to statistics, 26,447 vulnerabilities were disclosed in 2023, continuing the alarming trend of continuous growth in vulnerability numbers over the past seven years[1]. Static source code

---

[*]Corresponding author

[1]https://www.skyboxsecurity.com/resources/report/vulnerability-threat-trends-report-2023/

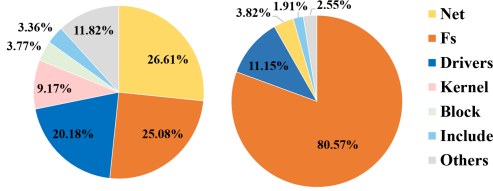

Figure 1: The distribution of CWE-416 (left) and CWE-119 (right) vulnerabilities across all modules in the Linux kernel over the past decade.

```
@@ -2134,9 +2135,11
@@ static int
netlink_dump(struct sock *sk)
{    ...
     nlk->cb_running = false;
+    module = cb->module;
+    skb = cb->skb;
     mutex_unlock(nlk->cb_mutex);
-    module_put(cb->module);
-    consume_skb(cb->skb);
+    module_put(module);
+    consume_skb(skb);
     return 0;
     ...
}
```

```
@@ -4255,9 +4258,8
@@static int
nft_set_desc_concat_parse(const
struct nlattr *attr,struct
nft_set_desc *desc)
{    ...
     len =
ntohl(nla_get_be32(tb[NFTA_SET_FIELD_
LEN]));
-    if (len * BITS_PER_BYTE / 32 >
         NFT_REG32_COUNT)
-        return -E2BIG;
+    if (!len || len > U8_MAX)
+        return -EINVAL;
     desc->field_len[desc
->field_count++] = len;
     ...
}
```

Figure 2: CWE-416 vulnerability (left) and CWE-119 vulnerability (right) discovered in the net module.

vulnerability detection (VD), as an integral part of the software development lifecycle, plays a crucial role in enhancing system security and building reliable, high-quality software systems. Early static analysis tools required experts to define scanning rules for VD. Machine learning (ML)-based VD methods, on the other hand, required manual predefinition of code features. Currently, deep learning (DL)-based VD methods can achieve automated VD without the need for manual feature definition and have proven effective in identifying potential vulnerability patterns. These studies can be categorized by the target of detection: (1) *Targeting open-source code projects*. Most of the studies like IVDetect [1] and Reveal[2] assess the performance of VD methods by evaluating their effectiveness across entire open-source projects, such as the Linux kernel, QEMU, and more. (2) *Focusing on specific vulnerability types*. Some studies primarily concentrate on specific commonly occurring vulnerability types in real-world applications, such as buffer overflow[3], use before initialization[4], use after free [5], and so on.

With the prevalent of software applications, the scale and complexity of source code projects increase. On the one hand, the functionalities of various components or modules within a project vary, leading to differences in the vulnerability triggering conditions and types. On the other hand, different triggering mechanisms result in a rich variety of subtypes within the same vulnerability type. Consequently, potential vulnerability patterns that may exist in target objects often closely correlate with specific source code contexts and characteristics of particular vulnerability types. However, the training objective of the current mainstream DL-based VD methods is to learn more comprehensive and effective vulnerability information, thereby obtaining an optimized model with the best generalization performance across the entire target source code dataset. In practical application, when a generlized model is applied to specific scope of source code, the detection of more detailed and distinguishable potential vulnerability patterns becomes difficult, leading to a compromise in the effectiveness of the model. Additionally, the vulnerability uncertainty and complexity of the target detection object, as well as the limitations of available vulnerability data, pose challenges for transfer learning technology in achieving effective knowledge transfer between the source and target domains. As a result, the generalization capability of the transfer model in the target domain will be severely limited. In security practice, scalable models capable of achieving robust VD performance across both the source and target domains are anticipated. To this end, it is essential to design specific pattern learning methods for different target tasks.

In this paper, we propose KF-GVD, a Knowledge Fusion-based GNN model for source code Vulnerability Detection. The object of KF-GVD is to implement the most suitable vulnerability pattern learning and identification strategy for task-oriented VD in a flexible manner, while maintaining a certain degree of generalization of the original model. By incorporating vulnerability knowledge into the feature learning process of the target, KF-GVD promotes the model to learn the vulnerability features that are closely related to the current task more efficiently. Through this way, the biases of the current general-purpose VD methods in target-oriented vulnerability pattern learning can be corrected to some extent. KF-GVD employs a graph self-attention mechanism to capture high-weight nodes that influence model decisions during message propagation. By associating with code statements, it obtains fine-grained vulnerability localization, achieving high transparency and interpretability of the model, thereby assisting developers in understanding the rationality of model decision-making. In summary, our contributions in this paper are:

- We propose a task-oriented knowledge fusion method, which integrates specific vulnerability knowledge in the feature learning process of GNN, it enables the general-purpose VD model to learn vulnerability features related to the current target more effectively and achieve VD tailored to specific tasks.

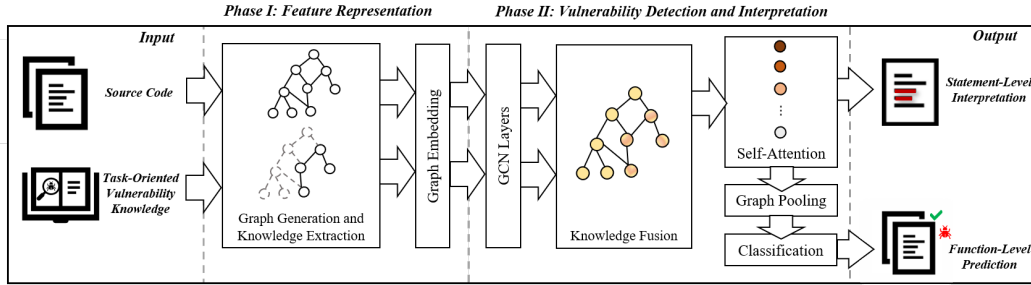

Figure 3: The overall architecture of KF-GVD.

- We propose the framework KF-GVD, which can be flexibly adapt to target task detection while maintaining the generalization performance for source task. The interpretable KF-GVD achieves effectively vulnerability identification at both the source code function level and statement level.

- Extensive experiments demonstrate the superiority of KF-GVD in VD performance compared to SOTAs when targeting multiple functional modules or diverse vulnerability subtypes. Note that KF-GVD discovered and submitted 9 undisclosed vulnerabilities in the open source C/C++ project without ground truth, proving its effectiveness in real-world applications.

## 2 Motivation

Figure 1 shows the distribution of the number of security commits related to CWE-416 (Use After Free) and CWE-119 (Buffer Overflow) in the Linux kernel over the past decade, collected from NVD. It can be observed that the occurrence proneness of different types of vulnerabilities exhibits a significant module tendency, especially CWE-119. Of the 11 modules in the Linux kernel, more than 80% of the vulnerabilities occur in the first four modules with the highest number of vulnerabilities. Additionally, distinct triggering features for different vulnerability types on the same module are evident. As shown in Figure 2, presenting two typical patch examples in the network module. It can be observed from the code snippet on the left, a race condition caused by pointer-related resource leakage release is a major triggering form of CWE-416 vulnerabilities in the net module. In contrast, CWE-119 in the same module are usually related to restrictive checks lacking critical network protocol fields.

The above examples are presented to demonstrate that in practical applications, the potential vulnerability patterns associated with program behavior differ and have distinct characteristics depending on the context of different detection targets and tasks. In such cases, the two existing types of DL-based VD methods, which employ uniform model optimization and analysis techniques for any target, are difficult to make full use of known information in diverse practical task scenarios to characterize the potential vulnerability characteristics of different target codes. The greatly compromised detection performance will also further burden the code audit in the software development phase. To alleviate the above problems, KF-GVD employs a simple and efficient knowledge guidance form of artificial "prompt", which enhances the rationality of model decision-making by combining the historical information of target tasks and human prior knowledge, corrects the deviation of pattern learning to a certain extent, and makes it more flexible to adapt to downstream tasks. As the first task-oriented GNN-based VD method, KF-GVD strives to achieve more accurate and scalable VD with lower costs.

## 3 The KF-GVD Framework

The overall architecture of KF-GVD is illustrated in Figure 3. KF-GVD consists of two main phases: feature representation and vulnerability detection and interpretation.

### 3.1 Feature Representation

In the feature representation phase, the source code is initially transformed into Code Property Graphs (CPGs) as the intermediate representation. Subsequently, task-specific vulnerability knowledge

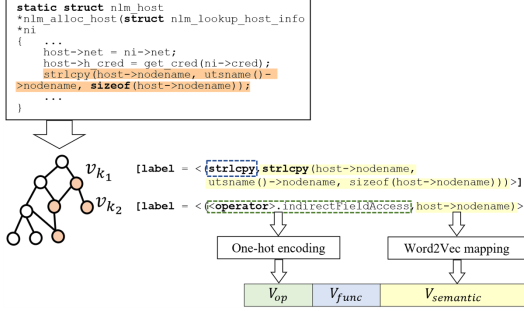

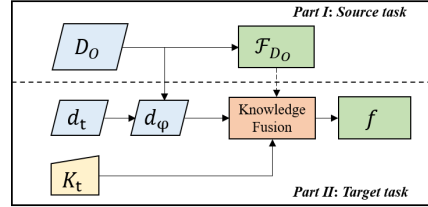

Figure 4: An example of feature representation.

Figure 5: The workflow of KF-GVD.

subgraphs are extracted from the CPGs. Finally, node feature vectors are encoded according to the node information of the graph, and combined with the adjacency matrix to form a graph embedding as the input of the model.

### 3.1.1 Code Property Graph Generation

The concept of CPG proposed by Yamaguchi, et al. (2014) [6] is a joint data structure that combines abstract syntax tree, control flow graph, and program dependency graph. CPG has been shown to model more diverse types of vulnerabilities than conducting a combination of single or two code properties as an intermediate representation of the source code for static analysis [7, 8]. The key insight behind KF-GVD's adoption of CPG as the intermediate representation is to comprehensively preserve source code information in this stage. This not only enhances the performance of the model in detecting vulnerabilities across the entire project, but also lays the foundation for the model to characterize more detailed vulnerability features in subsequent target tasks.

### 3.1.2 Task-oriented Vulnerability Knowledge Extraction

KF-GVD extracts the knowledge subgraph by initially screening the codes with the most feature relevance to the current task target, thus affecting the optimization from the source task generalization model to the specialized target expert model. The task-oriented vulnerability knowledge considered by KF-GVD includes: *vulnerable program operations* and *sensitive functions* related to the vulnerability type, as well as *customized knowledge for specific tasks* associated with the target functional scenarios. As for implementation, we first perform a coarse screening of source code statements for specific task objectives based on relevant vulnerability knowledge. Then, we map the identified statements to node sets in the CPG and mask out the remaining nodes weakly related to the task. The left part of the feature representation example in figure 4 shows the extraction process of the task-oriented vulnerability knowledge subgraph. For CWE-119 VD in the file module of the Linux kernel, because the code statement (highlighted in orange) in the code snippet calls the *strlcpy()* function, which is considered to be a sensitive function that may introduce CWE-119 type vulnerabilities under the current task, the associated node (highlighted in orange) and edges of this statement in CPG is extracted as the knowledge subgraph.

Appendix A provides a more detailed description of the task-oriented vulnerability knowledge considered by KF-GVD.

### 3.1.3 Graph Embedding

The CPG node information generated by Joern[2] comprises two parts: code element types and code statements. Figure 4 shows the corresponding information for nodes $v_{k1}$ and $v_{k2}$ in the CPG. KF-GVD adopts two vectors, $V_{op}$ and $V_{func}$, to describe the code element types. $V_{op}$ reflects the operation type of the code associated with the node, such as field access, memory allocation, mathematical operations, and so on. $V_{func}$ indicates special function calls, code field types, and so on. Both $V_{op}$ and $V_{func}$ use one-hot encoding, and when a node contains only one type of information, the remaining vector is zero-encoded. Next, we employ a pretrained Word2Vec model to map tokens

generated from code statements to fixed-length feature vectors to generate $V_{semantic}$ that represent the semantic information of the source code corresponding to the node. Finally, the three vectors are concatenated to form the final feature vector of the node. The spatial features of CPG represented by the adjacency matrix and the node feature matrix constitute the graph embedding, serving as the input for the subsequent model stage.

## 3.2 Vulnerability Detection and Interpretation

In this section, we provide a detailed description of the GNN model adopted in KF-GVD and explain how vulnerability knowledge is leveraged to optimize the model for specific subtasks during VD. Finally, we describe how KF-GVD utilizes attention mechanisms to interpret and locate potential vulnerability source code.

### 3.2.1 Object

For the source code function $C$ corresponding to the $CPG$ $G = (V, E), G \in \mathcal{G}$, we define $y \in \mathcal{Y}$ as the function-level label of $G$ corresponding to $C$, where $Y = \{0, 1\}$, $y = 1$ indicates that the sample is a vulnerability sample, $y = 0$ otherwise. We define source task O corresponding to dataset $D_O = \{(G_i, y_i)|G_i \in \mathcal{G}_O, y_i \in Y_O\}_{i=1}^N$. Target task T corresponds to dataset $D_T = \{(G_i, y_i)|G_i \in \mathcal{G}_T, y_i \in Y_T\}_{i=1}^M$, where $N$ and $M$ are the total number of samples contained in $D_O$ and $D_T$, and $N > M$. Assuming that a mapping $\mathcal{F}_{D_O} : \mathcal{G}_O \to \mathcal{Y}_O$ has been learned on the source task dataset $D_O$, KF-GVD aims to establish a mapping $f : \mathcal{G}_t \to Y_t$ on the target task $T$ with the assistance of knowledge $K_t$ relevant to task $t$. Here, $d_t$ represents the dataset associated with any subtask $t$ under the target task $T$. Therefore, the workflow of KF-GVD is mainly divided into two parts: model training on the source task and knowledge fusion on the target task. The workflow of KF-GVD is shown in Figure 5.

### 3.2.2 Task-oriented Model with Knowledge Fusion

The second phase in Figure 3 shows the general hierarchical structure of the GNN employed by KF-GVD. The model utilizes two layers of Graph Convolution Network (GCN) to calculate the new representation of each node by weighted summation of neighbor node information. Subsequently, based on the self-attention mechanism, the model obtains weights between nodes, calculating and retaining high-weighted nodes during message propagation [9]. The graph pooling layer is then employed to obtain the embedding of the entire graph. Finally, the model obtains function-level vulnerability identification results for the source code based on the classification layer. Furthermore, the training process of the model in the source task stage is similar to the current DL-based VD method. On this foundation, when conducting VD tasks for specific targets, a task-oriented knowledge fusion layer is introduced.

The current state of the graph $G = (V, E)$, obtained from the GCNs layer, is denoted as $H_V = \{h_{v_j}|v_j \in V\}, j \in \{1, .., n\}$, and the corresponding knowledge subgraph $G_k = (V_k, E_k)$ on the target task $t$ is denoted as $H_{V_k} = \{h_{v_q}|v_q \in V_k\}, q \in \{1, .., m\}$. The calculation of the state $H_u = \{h_{u_j}|v_j \in V\}, j \in \{1, ..., n\}$ for $G = (V, E)$ in the fusion layer can be defined as follows:

$$h_{u_j} = \begin{cases} Fusion(\alpha h_{v_j}, \beta h_{v_q}), v_j \in V_k \\ h_{v_j}, v_j \notin V_k \end{cases} \tag{1}$$

where $\alpha$ and $\beta$ are fusion coefficients, reflecting the degree to which knowledge tailored for a specific task $t$ influences the pattern learning of the model. Assuming that in the source task $\mathcal{O}$, the model has been trained on dataset $\mathcal{D}_\mathcal{O}$, denoted as $\mathcal{F}_{D_O}$. The training of model $f$ for a subtask $t, t \in \mathcal{T}$ includes the following steps:

(1) *Dataset collection*: Define task $t$ corresponding to dataset $d_t \in \mathcal{D}_\mathcal{T}$. To ensure that the fused model $f$ retains the ability to detect vulnerabilities in $\mathcal{O}$ to some extent, the training set $d_\varphi$ for $f$ is derived from $d_t$ and random sampling of $\mathcal{D}_\mathcal{O}$, defined as $d_o \in \mathcal{D}_\mathcal{O}, d_\varphi = d_o + d_t$.

(2) *Initialization*: Initialize the parameters of $f$ using $\mathcal{F}_{D_O}$ to achieve knowledge sharing between the source task and the target task.

(3) *Fusion*: During the training phase, we perform the feature fusion operation (Equation 1) only on the data $d'_t$ randomly sampled from $d_t$. The fusion function employs a weighted sum of node features.

For implementation, we first evaluate the detection performance of $\mathcal{F}_{D_O}$ on $d_t$ and consider the misclassified samples in the first round by $\mathcal{F}_{D_O}$ as the preferred subset of $d_t'$ for the fusion operation in the training process of $f$. We treat this subset of samples as the data subset with understanding bias of $\mathcal{F}_{D_O}$ in the target task. Overall, the knowledge fusion process employed by KF-GVD is also a process of manual correction and adjustment carried out during the model training phase.

A more detailed computational description of each layer of the model is provided in Appendix B.

### 3.2.3  Statement-level Interpretation and Location

KF-GVD achieves interpretability through the self-attention mechanism. In the self-attention layer, the top $\lceil \gamma n \rceil$ nodes with the highest attention scores in $G$ will be retained as the input for the pooling layer, and the states of the remaining nodes will be masked during subsequent message propagation, where $\gamma$ is the retain rate. The source code statements corresponding to the crucial nodes which have high influence on $f$ to make vulnerability decisions are regarded as the interpretation of the current sample, that is, the fine-grained vulnerability statement location. In the implementation, we realize the mapping from graph nodes to source code statements through the data files generated by joern.

To improve the performance of KF-GVD in function-level and statement-level VD, we define a graph $G = (V, E)$ with a labeled node set $\mathcal{N} = \{(v_j, l_j) | v_j \in V, l_j \in L\}_{j=1}^n$, where $l$ represents the node label. The prediction function $f$ for the target task $t$ can be learned by minimizing the following loss function:

$$\min \sum_{i=1}^{|d_\varphi|} [\mathcal{L}_{CE}(f(g_i), y_i) + \lambda \sum_{j=1}^{n} \mathcal{L}_{CE}(f(g_i), l_j | y_i))] \tag{2}$$

Here, $|d_\varphi|$ represents the size of the training dataset under task $t$, $\lambda$ is an adjustable hyperparameter.

## 4  Evaluation

We evaluate the superiority of KF-GVD compared to baseline approaches on two source tasks, $S_{119}$ and $S_{416}$, as well as their corresponding target tasks, $T_m$ and $T_{sub}$, in order to answer the following research questions:

**RQ1**: *How does KF-GVD perform compared to other function-level VD methods?*

**RQ2**: *How does the interpretation of KF-GVD compare to other fine-grained VD methods in terms of locating vulnerable statements?*

**RQ3**: *How does KF-GVD perform in detecting and locating vulnerabilities in real-world software products where the presence of vulnerabilities is unknown?*

### 4.1  Experiment Settings

#### 4.1.1  Dataset

The evaluation dataset consists of two parts: the source task dataset and the target task dataset. Following the current practice of DL-based VD methods, we initially train the model on a widely collected source task dataset to obtain a model with good generalization performance for a specific vulnerability type. Subsequently, the model is applied to the target task dataset for VD.

*Source Task Dataset*: We define two source tasks targeting the detection of CWE-119 and CWE-416 type vulnerabilities, denoted as $S_{119}$ and $S_{416}$ respectively. The source task dataset consists of 80% CWE-119 and CWE-416 type vulnerability information extensively collected from 13 real-world C++ projects from NVD[3]. The remaining 20% is sourced from academic security defects and synthetic data provided by SARD[4].

*Target Task Dataset*: Referring to the types of research objects in the current DL-based VD studies (Section 1), our target tasks are divided into two categories: (1) $T_m$: Detection of CWE-119 and CWE-416 type vulnerabilities within various modules in the Linux kernel, denoted as $T_{m_{119}}$ and $T_{m_{416}}$ respectively. (2) $T_{sub}$: Detection of CWE-119 subtypes, specifically CWE-125 and CWE-787.

Table 1: Comparison of function-level VD of $T_{m_{119}}$ and $T_{sub}$ on $S_{119}$. P: Precision(%); R: Recall(%); F1: F1-score(%)

| Method | $S_{119}$ | | | $T_{m_{119}}$ | | | | | | | | | | | | $T_{sub}$ | | | | | |
| | | | | Fs | | | Drivers | | | Net | | | Include | | | CWE-125 | | | CWE-787 | | |
| | P | R | F1 | P | R | F1 | P | R | F1 | P | R | F1 | P | R | F1 | P | R | F1 | P | R | F1 |
|---|---|---|---|---|---|---|---|---|---|---|---|---|---|---|---|---|---|---|---|---|---|
| Cppcheck | 45.0 | 55.7 | 49.8 | 33.7 | 50.5 | 40.4 | 32.1 | 45.9 | 37.8 | 44.2 | 40.0 | 42.0 | 23.9 | 35.7 | 28.6 | 24.8 | 50.6 | 33.3 | 29.4 | 35.7 | 32.2 |
| Flawfinder | 27.6 | 50.4 | 35.7 | 15.3 | 57.4 | 24.2 | 25.9 | 44.8 | 32.8 | 37.6 | 42.8 | 40.0 | 29.7 | 56.8 | 39.0 | 12.9 | 37.4 | 19.2 | 18.3 | 33.5 | 23.7 |
| Sysver | 54.8 | 70.6 | 61.7 | 23.6 | 67.2 | 34.9 | 28.3 | 56.2 | 37.6 | 15.7 | 60.9 | 25.0 | 33.0 | 42.6 | 37.2 | 39.7 | 58.4 | 47.3 | 33.4 | 48.6 | 39.6 |
| VulCNN | 63.9 | 77.4 | 70.0 | 35.5 | 50.7 | 41.8 | 27.8 | 44.6 | 34.3 | 39.4 | 58.6 | 47.1 | 22.0 | 43.5 | 29.2 | 16.8 | 29.1 | 21.3 | 17.6 | 33.0 | 23.0 |
| Codebert | 65.2 | 67.9 | 66.5 | 54.7 | 39.5 | 45.9 | 37.5 | 40.0 | 38.7 | 48.5 | 44.1 | 46.2 | 34.6 | 51.8 | 41.5 | 34.8 | 57.6 | 43.4 | 43.7 | 48.6 | 46.0 |
| CodeLlama | 70.0 | 64.1 | 66.9 | 55.7 | 54.9 | 55.3 | 45.6 | 45.8 | 45.7 | 57.2 | 48.0 | 52.2 | 49.3 | 53.9 | 51.5 | 37.6 | 53.9 | 44.3 | 48.0 | 55.8 | 51.6 |
| Wizardcoder | 72.4 | 52.4 | 60.8 | 62.5 | 35.5 | 45.3 | 45.8 | 48.7 | 47.2 | 50.8 | 42.0 | 46.0 | 48.6 | 56.6 | 52.3 | 33.5 | 52.5 | 40.9 | 47.5 | 51.9 | 49.6 |
| Devign | 68.5 | 70.2 | 69.3 | 30.6 | 54.2 | 39.1 | 35.4 | 42.8 | 38.7 | 48.6 | 57.2 | 52.6 | 25.8 | 40.3 | 31.5 | 20.1 | 37.9 | 26.3 | 18.4 | 25.0 | 21.2 |
| ReGVD | 74.1 | 71.2 | 72.6 | 60.8 | 34.2 | 43.8 | 40.9 | 47.1 | 43.8 | 52.1 | 59.1 | 55.4 | 44.1 | 50.8 | 47.2 | 29.8 | 54.0 | 38.4 | 44.9 | 57.2 | 50.3 |
| IVDetect | 79.0 | 83.3 | 81.1 | 46.7 | 33.3 | 38.9 | 33.3 | 66.7 | 44.4 | 66.7 | 50.0 | 57.1 | 40.0 | 46.2 | 42.9 | 31.9 | 55.8 | 38.1 | 46.8 | 52.4 | 43.0 |
| GVD-ft | 82.9 | 90.9 | 86.7 | 73.5 | 58.7 | 65.2 | 66.7 | 88.9 | 76.2 | 76.3 | 58.5 | 64.7 | 57.1 | 61.5 | 59.3 | 49.8 | 60.5 | 54.6 | 66.7 | 61.5 | 64.0 |
| **KF-GVD** | 82.9 | 90.9 | 86.7 | 96.1 | 95.2 | 95.7 | 90.0 | 94.7 | 92.3 | 91.7 | 75.0 | 82.5 | 91.7 | 84.6 | 88.0 | 59.2 | 80.0 | 67.9 | 80.0 | 84.2 | 82.1 |

Table 2: Comparison of function-level VD of $T_{m_{416}}$ on $S_{416}$. P: Precision(%); R: Recall(%); F1: F1-score(%)

| Method | $S_{416}$ | | | $T_{m_{416}}$ | | | | | | | | | | | | | | | | | |
| | | | | Net | | | Fs | | | Drivers | | | Kernel | | | Block | | | Include | | |
| | P | R | F1 | P | R | F1 | P | R | F1 | P | R | F1 | P | R | F1 | P | R | F1 | P | R | F1 |
|---|---|---|---|---|---|---|---|---|---|---|---|---|---|---|---|---|---|---|---|---|---|
| Cppcheck | 27.7 | 42.6 | 33.6 | 14.8 | 22.7 | 17.9 | 27.0 | 53.6 | 35.9 | 30.7 | 45.9 | 36.8 | 10.3 | 45.9 | 16.8 | 30.2 | 36.5 | 33.1 | 23.6 | 31.8 | 27.1 |
| Flawfinder | 33.4 | 45.9 | 38.7 | 20.6 | 36.6 | 26.4 | 15.9 | 42.6 | 23.2 | 5.6 | 22.4 | 9.0 | 28.5 | 62.8 | 39.2 | 17.8 | 26.7 | 21.4 | 25.0 | 39.7 | 30.7 |
| Sysver | 58.4 | 67.2 | 62.5 | 21.9 | 40.5 | 28.4 | 27.2 | 37.3 | 31.5 | 32.5 | 30.7 | 31.6 | 22.7 | 23.6 | 23.1 | 37.9 | 30.2 | 33.6 | 26.3 | 45.7 | 33.4 |
| VulCNN | 66.9 | 72.8 | 69.7 | 33.4 | 52.7 | 40.9 | 47.0 | 52.4 | 49.6 | 28.5 | 43.1 | 34.3 | 36.8 | 56.3 | 44.5 | 24.7 | 65.1 | 35.8 | 22.5 | 39.6 | 28.7 |
| Codebert | 66.2 | 62.3 | 64.2 | 50.3 | 42.2 | 45.9 | 47.8 | 36.7 | 41.5 | 42.3 | 51.6 | 46.5 | 46.5 | 51.1 | 48.7 | 42.9 | 40.8 | 41.8 | 40.9 | 35.1 | 37.8 |
| CodeLlama | 65.9 | 59.1 | 62.3 | 52.9 | 46.0 | 49.2 | 50.6 | 52.6 | 51.6 | 44.1 | 43.9 | 44.0 | 53.3 | 51.5 | 52.4 | 40.5 | 41.1 | 40.8 | 57.6 | 52.8 | 55.1 |
| Wizardcoder | 59.6 | 69.3 | 64.1 | 53.7 | 48.2 | 50.8 | 42.7 | 38.2 | 40.3 | 39.8 | 52.0 | 45.1 | 55.6 | 50.5 | 52.9 | 44.3 | 38.5 | 41.2 | 56.4 | 49.3 | 52.6 |
| Devign | 63.7 | 79.4 | 70.7 | 37.1 | 42.6 | 39.7 | 48.9 | 50.2 | 49.5 | 34.1 | 56.9 | 42.6 | 37.5 | 44.6 | 40.7 | 48.1 | 33.9 | 39.8 | 36.8 | 70.4 | 48.3 |
| ReGVD | 67.2 | 71.7 | 69.4 | 41.9 | 43.7 | 42.8 | 50.3 | 51.5 | 50.9 | 40.6 | 45.9 | 43.1 | 45.8 | 55.5 | 50.2 | 42.8 | 34.8 | 38.4 | 44.5 | 65.2 | 52.9 |
| IVDetect | 81.8 | 94.7 | 87.8 | 40.4 | 36.2 | 38.2 | 51.7 | 51.9 | 51.8 | 43.8 | 41.2 | 42.4 | 41.4 | 60.0 | 49.0 | 39.0 | 35.6 | 37.2 | 66.7 | 80.0 | 72.7 |
| GVD-ft | 86.8 | 89.3 | 88.0 | 53.6 | 88.2 | 66.7 | 73.3 | 66.0 | 69.5 | 41.3 | 44.7 | 42.9 | 39.5 | 44.7 | 42.0 | 51.7 | 78.4 | 51.9 | 50.0 | 53.3 | 51.6 |
| **KF-GVD** | 86.8 | 89.3 | 88.0 | 78.6 | 98.1 | 87.3 | 73.9 | 94.4 | 82.9 | 87.1 | 71.8 | 78.7 | 85.4 | 92.1 | 88.6 | 66.7 | 83.3 | 74.1 | 82.4 | 93.3 | 87.5 |

Both of these vulnerability subtypes are also included in the 2023 CWE Top 25 Most Dangerous Software Weaknesses[5].

### 4.1.2 Baseline Approaches and Evaluation Metrics

We compare KF-GVD at the function level with four types of VD methods, including: (1)*Rule-based commercial static code analysis tools*: Cppcheck[6] and Flawfinder[7]. (2) *Classical DL-based VD methods*: Sysver [10] and VulCNN [11]. (3) *Large-scale code models*: Codebert [12], Code Llama [13], and Wizardcoder [14]. (4) *GNN-based VD methods*: Devign [8], ReGVD [15] and IVDetect [1]. For statement-level localization comparison, we chose the state-of-the-art fine-grained VD methods IVDetect [1], LineVul [16] and LineVD [17] as baselines.

We evaluate the performance of KF-GVD and the baseline methods using Precision (P), Recall (R), and F1-score (F) for both function-level and statement-level comparison. Besides, we also introduce the ranking metric Mean Average Precision (MAP) in order to explore the performance of the method on code statements that it considers as the best interpretation for function prediction. Appendix C provides more specific information on the datasets and experimental settings adopted by KF-GVD.

### 4.2 Function-level Vulnerability Detection Performance

For ML-based VD methods in addition to large-scale code models, we train and test all models on the source tasks $S_{119}$ and $S_{416}$ separately, and then apply them to the corresponding target tasks. Additionally, following the concept of transfer learning, we employed the same GNN model as KF-GVD in the source tasks. For VD in the target tasks, we fine-tuned the GNN model used in KF-GVD, denoted as GVD-ft, instead of employing knowledge fusion operations, so the results of VD on the source task are the same as KF-GVD.

Table 3: Comparison of statement-level VD of $T_{m_{119}}$ and $T_{sub}$ on $S_{119}$. P: Precision(%); R:Recall(%); F1: F1-score(%)

| Method | | $T_{m_{119}}$ | | | | | | | | | | | | $T_{sub}$ | | | | |
| | Fs | | | Drivers | | | Net | | | Include | | | CWE-125 | | | CWE-787 | | |
| | P | R | F | P | R | F | P | R | F | P | R | F | P | R | F | P | R | F |
|---|---|---|---|---|---|---|---|---|---|---|---|---|---|---|---|---|---|---|
| IVDetect | 32.3 | 56.1 | 34.8 | 10.5 | 63.1 | 15.4 | 36.7 | 20.4 | 26.0 | 9.7 | 74.7 | 16.4 | 2.2 | 17.1 | 3.1 | 16.7 | 10.0 | 12.5 |
| LineVD | 39.2 | 27.9 | 32.6 | 11.0 | 58.7 | 16.1 | 37.6 | 21.2 | 26.8 | 17.2 | 53.2 | 26.1 | 4.1 | 24.9 | 5.3 | 33.3 | 20.0 | 25.0 |
| LineVul | 33.8 | 45.0 | 38.6 | 10.7 | 24.0 | 14.8 | 22.4 | 28.0 | 24.9 | 16.3 | 44.8 | 23.9 | 6.4 | 13.6 | 8.7 | 29.8 | 19.0 | 23.2 |
| GVD-ft | 32.1 | 55.0 | 34.5 | 11.2 | 66.0 | 16.4 | 18.3 | 10.2 | 13.0 | 9.6 | 85.4 | 16.3 | 7.5 | 51.0 | 10.3 | 2.9 | 1.8 | 2.2 |
| **KF-GVD** | 82.1 | 58.7 | 66.6 | 38.2 | 81.1 | 49.6 | 74.7 | 65.5 | 66.3 | 54.9 | 84.4 | 65.0 | 31.9 | 55.8 | 38.1 | 29.2 | 67.9 | 31.4 |

Table 4: Comparison of statement-level VD of $T_{m_{416}}$ on $S_{416}$. P: Precision(%); R: Recall(%); F1: F1-score(%)

| Method | | | | | | | | | $T_{m_{416}}$ | | | | | | | | | |
| | Net | | | Fs | | | Drivers | | | Kernel | | | Block | | | Include | | |
| | P | R | F | P | R | F | P | R | F | P | R | F | P | R | F | P | R | F |
|---|---|---|---|---|---|---|---|---|---|---|---|---|---|---|---|---|---|---|
| IVDetect | 19.6 | 58.1 | 24.5 | 15.4 | 80.8 | 19.2 | 20.2 | 77.9 | 25.5 | 27.7 | 83.8 | 36.9 | 15.4 | 23.6 | 18.6 | 67.9 | 67.7 | 67.5 |
| LineVD | 24.0 | 98.8 | 31.3 | 16.6 | 55.9 | 25.6 | 17.9 | 75.2 | 23.3 | 15.8 | 72.9 | 21.9 | 12.5 | 16.7 | 14.3 | 48.2 | 49.2 | 48.7 |
| LineVul | 20.9 | 45.3 | 28.6 | 15.3 | 44.0 | 22.7 | 22.8 | 32.8 | 26.9 | 24.9 | 41.8 | 31.2 | 14.1 | 48.0 | 21.8 | 31.7 | 36.4 | 33.9 |
| GVD-ft | 22.7 | 58.6 | 25.3 | 16.7 | 71.3 | 21.8 | 25.3 | 69.9 | 28.0 | 16.4 | 66.5 | 22.4 | 10.8 | 55.3 | 15.2 | 52.9 | 52.2 | 52.4 |
| **KF-GVD** | 56.3 | 96.3 | 63.8 | 55.9 | 80.8 | 66.0 | 76.5 | 81.1 | 68.1 | 80.6 | 75.9 | 75.1 | 27.4 | 97.3 | 36.1 | 73.3 | 73.1 | 72.6 |

Table 1 and Table 2 show the comparison of function-level VD performance on different source tasks and their corresponding target tasks. It can be obviously observed that despite all baseline methods being designed to detect the same type of vulnerabilities, they exhibit a noticeable decrease in performance metrics across various specific target tasks. In contrast, KF-GVD consistently performs well. Compared to the best results among baseline methods (indicated by underlines in the table), KF-GVD demonstrates an improvement in precision by 0.6%-44%, recall by 5.8%-29.3%, and an average gain of 22.6% on F1-score. Although GVD-ft achieved relatively better results than other baseline methods in most target tasks due to fine-tuning of the model, GVD-ft shows a 23.3% lower F1 average on $T_{m_{119}}$, a 31.6% lower F1-score average on $T_{m_{416}}$, and a 15.7% lower F1-score average on $T_{sub}$ compared to KF-GVD. This further proves the flexibility and effectiveness of our approach in VD tailored to specific tasks.

Overall, the effectiveness of rule-based static analysis tools is inferior to that of ML-based VD methods. This is due to the limitations of finite and fixed manually predefined static scanning rules when dealing with multiple types of vulnerabilities and detection targets. Moreover, methods that adopt models like LSTM and CNN exhibit an average reduction of 20.6% on F1-score compared to approaches based on GNNs. To some extent, this reflects the powerful spatial feature learning capability of GNNs when dealing with structured languages like source code, as opposed to the flattened feature processing approach of classical DL networks. It is worth noting that the emerging large-scale code models have mediocre performance in dealing with tasks related to code vulnerability detection, and the F1-scores on the corresponding target tasks of $S_{119}$ and $S_{416}$ are on average 34.3% and 33.4% lower than those of KF-GVD. We infer that this is because current large language models mainly focus on question and answer, completion and other generative tasks rather than classification tasks.

### 4.3 Statement-level Vulnerability Detection Performance

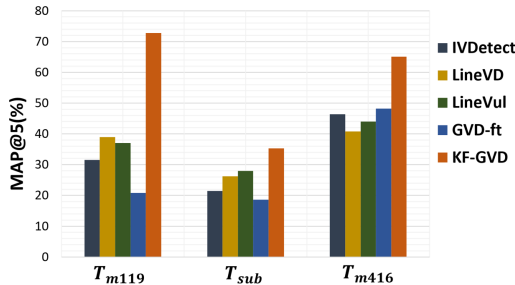

Figure 6: Statement-level VD comparison on MAP@5.

Table 3 and Table 4 present the comparison of statement-level vulnerability localization results between KG-GVD and other baseline methods on the target tasks corresponding to $S_{119}$ and $S_{416}$. In three different target tasks, KF-GVD exhibits a precision improvement of 5.4% - 52.9% and an average recall enhancement of 2.6% - 73.7% in statement-level vulnerability localization compared to suboptimal detection results. This corresponds to an average gain of 59.7% in precision and 30.9% in recall, showcasing KF-GVD's robust coverage and localization capabilities for vulnerable code during fine-grained detection. It is worth noting that GVD-ft, which

performs suboptimally at the function level, does not exhibit equally good results at the statement level. This indicates that the fine-tuning approach employed by GVD-ft does not accurately capture the underlying vulnerability patterns of the target task. Moreover, MAP reflects the average precision on the top K confident predictions. Following established research practices [1, 18], we set K=5 and compare the statement-level MAP across all methods, as illustrated in Figure 6. Observably, KF-GVD consistently outperforms other methods in MAP on all target tasks, particularly achieving an average improvement of 42.4% on $T_{m119}$, which further proving the rationality of the function-level predictions from KF-GVD.

## 4.4 Case Study

We applied KF-GVD to various C++ open-source libraries commonly targeted for vulnerability detection. For projects without ground truth, we initially gathered information on publicly disclosed vulnerabilities for the target objects (obtained from GitHub) and performed detection for specific unknown vulnerabilities, particularly focusing on several high-risk issues. Taking multiple versions of Assimp (Open Asset Import Library), used for importing various 3D model file formats, as an example, we conducted detection on a total of 312 C++ source code files under the Assimp's assetlib directory. The detection revealed 19 vulnerabilities, including three reported and published vulnerabilities (CVE-2022-38528, CVE-2022-45748, CVE-2021-45948), 3 pending confirmation security issues, and 2 undisclosed vulnerabilities that we have submitted to CNNVD and been confirmed. Table 5 presents all the undisclosed vulnerabilities detected by KF-GVD in different C++ open-source objects.

Table 5: Undisclosed vulnerabilities detected by KF-GVD in different C++ open-source objects.

| ID | Project | File Location | Vul_line |
|---|---|---|---|
| CNNVD-2023-43767151 | assimp | /.../OpenDDLParser.cpp | 348 |
| CNNVD-2023-12599427 | | /.../FBXParser.cpp | 192 |
| CNNVD-2023-59936877 | boost | /.../detail/rapidxml.hpp | 644 |
| CNNVD-2023-23489133 | | /.../basic_regex_creator.hpp | 710 |
| CNNVD-2023-20301510 | c-blosc2 | /.../blosc-private.h | 120 |
| CNNVD-2023-76730942 | exiv2 | /.../value.cpp | 13 |
| CNNVD-2023-90736138 | flatbuffers | /.../util.h | 133 |
| CNNVD-2023-83881569 | frr | /.../bgp_attr.c | 2658 |
| CNNVD-2023-27702356 | harfbuzz | /.../hb-atomic.hh | 172 |

## 5 Disscusion

### 5.1 Threats to Validity

First, in practical applications, we find that KF-GVD has more obvious advantages in target tasks with more historical version iteration information and knowledge related to specific vulnerability types, which indicates that the performance of the method can be continuously improved with the increase of target task-related vulnerability knowledge. However, without any historical vulnerability information about the target software or knowledge about a specific vulnerability type, method performance degrades to the level of vulnerability detection with the generalized model. Second, we only verify the validity of the method on C/C++ code, not other programming languages, and in principle the method can be extended. Third, we only focus on vulnerability identification and fine-grained localization within functions, and cannot detect source code vulnerabilities that have cross-function or cross-file dependencies.

### 5.2 Limitations and Future Work

During the data processing stage, generating CPG data from source files accounts for more than 75% of the total processing time, highlighting the need for more efficient data generation tools and processing strategies in the future. Additionally, in the feature embedding stage, we truncate the $V_{code}$ that exceeds the length threshold of the feature vector, which leads to the loss of semantic information to a certain extent. In the next phase, leveraging the powerful representation and generation capabilities

of current large-scale code models to obtain more comprehensive and efficient source code features will be a future research direction to enhance vulnerability localization performance. Finally, while KF-GVD has significantly improved statement-level vulnerability localization compared to SOTAs, there remains substantial room for improvement in fine-grained localization precision relative to function-level detection results.

More ablation studies on vulnerability knowledge sensitivity, feature representation, and experiments on cross-domain tasks are provided in Appendix D.

## 6    Related Works

Current DL-based VD methods achieve automated VD, alleviating the manual burden associated with rule-based [19] and ML-based approaches [20, 21, 22, 23, 24]. VulDeePecker [3], Sysevr [10], $\mu$VulDeePecker [25], VulCNN [11] combined with classical DL models such as LSTM, BGRU, CNN, are employed to perform VD on various open-source projects or specific vulnerability types. The detection granularity of these methods is at the slice or function level. Additionaly, Devign [8], BGNN4VD [26], Reveal [2] and many other studies based on GNNs perform function-level VD on entire projects such as QEMU, FFmpeg, Linux kernel, or mixed datasets.

In recent years, some studies have achieved fine-grained vulnerability localization. VulDeelocator [27] utilizes intermediate code to define program segments for VD, accommodating semantic information that cannot be conveyed by source code-based representations. By employing the idea of granularity refinement, VulDeelocator outputs a finer granularity than its input, enhancing the precision of the detector. Furthermore, IVDetect [1] considers vulnerable statements and their surrounding context separately through data dependency and control dependency, enabling the model to better distinguish vulnerable statements. Additionally, IVDetect introduces GNNExplainer [28] to provide subgraphs in program dependency graph (PDG) as explanations, containing key statements related to the detected vulnerabilities. Besides, LineVD [18] defines statement-level vulnerability detection as a node classification task, utilizing GNNs to leverage control and data dependency relationships between statements. By resolving conflicts between function-level and statement-level information and learning from both levels, LineVD significantly improves performance.

## 7    Conclusion

In this paper, we propose KF-GVD, a knowledge fusion-based vulnerability detection method. KF-GVD alleviates the limitations of current general-purpose detection methods when applied to the contexts involving multiple functional modules or diverse types of vulnerabilities. By integrating task-oriented vulnerability knowledge, KF-GVD prompts the model to efficiently explore vulnerabilitiy patterns tailored to specific tasks while still maintaining general task performance. Our empirical evaluations demonstrate the superior performance of KF-GVD tailored for diverse specific tasks in both function-level and statement-level VD. The case study we conducted on real C++ open-source projects further substantiates the practical effectiveness of KF-GVD in real-world applications. Notably, KF-GVD identified 9 undisclosed vulnerabilities when applied to real-world C++ open-source projects, further proving its practicality.

## Footnotes

[2]https://joern.io/

[3]https://nvd.nist.gov/

[4]https://samate.nist.gov/SARD/test-suites

[5]https://cwe.mitre.org/data/definitions/1425.html

[6]http://cppcheck.net

[7]https://dwheeler.com/flawfinder

## References

[1] Yi Li, Shaohua Wang, and Tien N. Nguyen. Vulnerability detection with fine-grained interpretations. In *Proceedings of the 29th ACM Joint Meeting on European Software Engineering Conference and Symposium on the Foundations of Software Engineering*, ESEC/FSE 2021, page 292–303, New York, NY, USA, 2021. Association for Computing Machinery.

[2] Saikat Chakraborty, Rahul Krishna, Yangruibo Ding, and Baishakhi Ray. Deep learning based vulnerability detection: Are we there yet. *IEEE Transactions on Software Engineering*, pages 1–1, 2021.

[3] Zhen Li, Deqing Zou, Shouhuai Xu, Xinyu Ou, Hai Jin, Sujuan Wang, Zhijun Deng, and Yuyi Zhong. Vuldeepecker: A deep learning-based system for vulnerability detection. *arXiv preprint arXiv:1801.01681*, 2018.

[4] Yizhuo Zhai, Yu Hao, Hang Zhang, Daimeng Wang, Chengyu Song, Zhiyun Qian, Mohsen Lesani, Srikanth V. Krishnamurthy, and Paul Yu. Ubitect: a precise and scalable method to detect use-before-initialization bugs in linux kernel. In *Proceedings of the 28th ACM Joint Meeting on European Software Engineering Conference and Symposium on the Foundations of Software Engineering*, ESEC/FSE 2020, page 221–232, New York, NY, USA, 2020. Association for Computing Machinery.

[5] Jia-Ju Bai, Julia Lawall, Qiu-Liang Chen, and Shi-Min Hu. Effective static analysis of concurrency Use-After-Free bugs in linux device drivers. In *2019 USENIX Annual Technical Conference (USENIX ATC 19)*, pages 255–268, Renton, WA, July 2019. USENIX Association.

[6] Fabian Yamaguchi, Nico Golde, Daniel Arp, and Konrad Rieck. Modeling and discovering vulnerabilities with code property graphs. In *2014 IEEE Symposium on Security and Privacy*, pages 590–604, 2014.

[7] Fabian Yamaguchi. Pattern-based vulnerability discovery. 2015.

[8] Yaqin Zhou, Shangqing Liu, Jingkai Siow, Xiaoning Du, and Yang Liu. Devign: Effective vulnerability identification by learning comprehensive program semantics via graph neural networks. *Advances in neural information processing systems*, 32, 2019.

[9] Junhyun Lee, Inyeop Lee, and Jaewoo Kang. Self-attention graph pooling. In *International conference on machine learning*, pages 3734–3743. PMLR, 2019.

[10] Zhen Li, Deqing Zou, Shouhuai Xu, Hai Jin, Yawei Zhu, and Zhaoxuan Chen. Sysevr: A framework for using deep learning to detect software vulnerabilities. *IEEE Transactions on Dependable and Secure Computing*, 2021.

[11] Yueming Wu, Deqing Zou, Shihan Dou, Wei Yang, Duo Xu, and Hai Jin. Vulcnn: an image-inspired scalable vulnerability detection system. In *Proceedings of the 44th International Conference on Software Engineering*, ICSE '22, page 2365–2376, New York, NY, USA, 2022. Association for Computing Machinery.

[12] Zhangyin Feng, Daya Guo, Duyu Tang, Nan Duan, Xiaocheng Feng, Ming Gong, Linjun Shou, Bing Qin, Ting Liu, Daxin Jiang, et al. Codebert: A pre-trained model for programming and natural languages. *arXiv preprint arXiv:2002.08155*, 2020.

[13] Wenhan Xiong Grattafiori, Alexandre Défossez, Jade Copet, Faisal Azhar, Hugo Touvron, Louis Martin, Nicolas Usunier, Thomas Scialom, and Gabriel Synnaeve. Code llama: Open foundation models for code. *arXiv preprint arXiv:2308.12950*, 2023.

[14] Ziyang Luo, Can Xu, Pu Zhao, Qingfeng Sun, Xiubo Geng, Wenxiang Hu, Chongyang Tao, Jing Ma, Qingwei Lin, and Daxin Jiang. Wizardcoder: Empowering code large language models with evol-instruct. *arXiv preprint arXiv:2306.08568*, 2023.

[15] Van-Anh Nguyen, Dai Quoc Nguyen, Van Nguyen, Trung Le, Quan Hung Tran, and Dinh Phung. Regvd: revisiting graph neural networks for vulnerability detection. In *Proceedings of the ACM/IEEE 44th International Conference on Software Engineering: Companion Proceedings*, ICSE '22, page 178–182, New York, NY, USA, 2022. Association for Computing Machinery.

[16] Michael Fu and Chakkrit Tantithamthavorn. Linevul: A transformer-based line-level vulnerability prediction. In *Proceedings of the 19th International Conference on Mining Software Repositories*, pages 608–620, 2022.

[17] David Hin, Andrey Kan, Huaming Chen, and M. Ali Babar. Linevd: statement-level vulnerability detection using graph neural networks. In *Proceedings of the 19th International Conference on Mining Software Repositories*, MSR '22, page 596–607, New York, NY, USA, 2022. Association for Computing Machinery.

[18] David Hin, Andrey Kan, Huaming Chen, and M Ali Babar. Linevd: Statement-level vulnerability detection using graph neural networks. *arXiv preprint arXiv:2203.05181*, 2022.

[19] Morteza Zakeri-Nasrabadi, Saeed Parsa, Mohammad Ramezani, Chanchal Roy, and Masoud Ekhtiarzadeh. A systematic literature review on source code similarity measurement and clone detection: Techniques, applications, and challenges. *Journal of Systems and Software*, 204:111796, 2023.

[20] Boris Chernis and Rakesh Verma. Machine learning methods for software vulnerability detection. In *Proceedings of the Fourth ACM International Workshop on Security and Privacy Analytics*, IWSPA '18, page 31–39, New York, NY, USA, 2018. Association for Computing Machinery.

[21] Guoyan Huang, Yazhou Li, Qian Wang, Jiadong Ren, Yongqiang Cheng, and Xiaolin Zhao. Automatic classification method for software vulnerability based on deep neural network. *IEEE Access*, 7:28291–28298, 2019.

[22] Riccardo Scandariato, James Walden, Aram Hovsepyan, and Wouter Joosen. Predicting vulnerable software components via text mining. *IEEE Transactions on Software Engineering*, 40(10):993–1006, 2014.

[23] Bo Shuai, Haifeng Li, Mengjun Li, Quan Zhang, and Chaojing Tang. Automatic classification for vulnerability based on machine learning. In *2013 IEEE International Conference on Information and Automation (ICIA)*, pages 312–318, 2013.

[24] Istehad Chowdhury and Mohammad Zulkernine. Using complexity, coupling, and cohesion metrics as early indicators of vulnerabilities. *Journal of Systems Architecture*, 57(3):294–313, 2011. Special Issue on Security and Dependability Assurance of Software Architectures.

[25] Deqing Zou, Sujuan Wang, Shouhuai Xu, Zhen Li, and Hai Jin. $\mu$vuldeepecker: A deep learning-based system for multiclass vulnerability detection. *IEEE Transactions on Dependable and Secure Computing*, 18(5):2224–2236, 2021.

[26] Sicong Cao, Xiaobing Sun, Lili Bo, Ying Wei, and Bin Li. Bgnn4vd: constructing bidirectional graph neural-network for vulnerability detection. *Information and Software Technology*, 136:106576, 2021.

[27] Zhen Li, Deqing Zou, Shouhuai Xu, Zhaoxuan Chen, Yawei Zhu, and Hai Jin. Vuldeelocator: a deep learning-based fine-grained vulnerability detector. *IEEE Transactions on Dependable and Secure Computing*, 2021.

[28] Zhitao Ying, Dylan Bourgeois, Jiaxuan You, Marinka Zitnik, and Jure Leskovec. Gnnexplainer: Generating explanations for graph neural networks. In H. Wallach, H. Larochelle, A. Beygelzimer, F. d'Alché-Buc, E. Fox, and R. Garnett, editors, *Advances in Neural Information Processing Systems*, volume 32. Curran Associates, Inc., 2019.

[29] Thomas N Kipf and Max Welling. Semi-supervised classification with graph convolutional networks. *arXiv preprint arXiv:1609.02907*, 2016.

[30] Xu Duan, Jingzheng Wu, Shouling Ji, Zhiqing Rui, Tianyue Luo, Mutian Yang, and Yanjun Wu. Vulsniper: Focus your attention to shoot fine-grained vulnerabilities. In *IJCAI*, pages 4665–4671, 2019.

[31] Vinod Nair and Geoffrey E Hinton. Rectified linear units improve restricted boltzmann machines. In *Icml*, 2010.

[32] Petar Veličković, Guillem Cucurull, Arantxa Casanova, Adriana Romero, Pietro Lio, and Yoshua Bengio. Graph attention networks. *arXiv preprint arXiv:1710.10903*, 2017.

[33] Guanjun Lin, Jun Zhang, Wei Luo, Lei Pan, Yang Xiang, Olivier De Vel, and Paul Montague. Cross-project transfer representation learning for vulnerable function discovery. *IEEE Transactions on Industrial Informatics*, 14(7):3289–3297, 2018.

[34] Zimin Chen, Steve Kommrusch, and Martin Monperrus. Neural transfer learning for repairing security vulnerabilities in c code. *IEEE Transactions on Software Engineering*, 49(1):147–165, 2023.

[35] Ashita Diwan, Miles Q. Li, and Benjamin C. M. Fung. Vdgraph2vec: Vulnerability detection in assembly code using message passing neural networks. In M. Arif Wani, Mehmed M. Kantardzic, Vasile Palade, Daniel Neagu, Longzhi Yang, and Kit Yan Chan, editors, *21st IEEE International Conference on Machine Learning and Applications, ICMLA 2022, Nassau, Bahamas, December 12-14, 2022*, pages 1039–1046. IEEE, 2022.

[36] Jiao Yin, MingJian Tang, Jinli Cao, and Hua Wang. Apply transfer learning to cybersecurity: Predicting exploitability of vulnerabilities by description. *Knowledge-Based Systems*, 210:106529, 2020.

# A    Task-Oriented Vulnerability Knowledge

In general, the task-oriented vulnerability knowledge employed by KF-GVD is derived from the matching rules of existing static analysis tools, relevant academic research findings, publicly available historical vulnerability information, and the expertise of security audit professionals. They are closely associated with the detected vulnerability types and contribute to the characterization of features relevant to the target tasks. Taking CWE-119 type vulnerabilities as an example, the task-oriented vulnerability knowledge considered by KF-GVD includes program operations and sensitive functions related to the specific vulnerability type, as well as vulnerability knowledge associated with the specific detection target, as follows:

***Vulnerable program operations***: The triggering of vulnerabilities is often caused by certain typical risky behaviors in the code. According to our statistics, approximately 46% of CWE-119 vulnerabilities in the Linux kernel are attributed to a lack of boundary checks (as shown on the right side of Figure 2). Other contributing factors include the absence of input-output validation, resource consumption due to memory recursion or iteration, and more.

***Sensitive functions***: Sensitive functions are closely related to hazardous operations within a program. For CWE-119 vulnerabilities, we collect and predefine over 80 relevant C/C++ language sensitive functions, including string copy, concatenation functions (e.g., strcpy, strncpy, strcat, strncat), input functions (e.g., gets, scanf, fscanf), dynamic memory allocation functions (e.g., alloca) and so on.

***Customized knowledge for specific tasks***: The manifestation of the same vulnerability type varies across different functional modules. For example, considering the two most frequently affected modules, fs and net, as illustrated in Figure 1, vulnerabilities of CWE-119 type are most prevalent in code statements related to path parsing, symbolic links, and file access within the file system module. In the network module, key areas introducing CWE-119 vulnerabilities include network protocol stack parsing, network data processing, and so on. As a result, based on the collected historical vulnerability information and considering the functional characteristics of the target tasks, task-related vulnerability nodes can be marked.

# B    KF-GVD Model

The computation for each layer of the KF-GVD model can be denoted as follows:

***GCN layer***: After graph embedding, the model input is $g = (X, A)$, where $X$ is the node feature matrix corresponding to $G$ and $A$ is the adjacency matrix corresponding to graph $G$. SAGPool uses a two-layer GCN structure for graph message propagation:

$$H_V = \sigma(GCN\_Layers(g(X, A))) \tag{3}$$

Where $\sigma(.)$ is the activation function, $H_V \in \mathbb{R}^{n \times d}$, where $d$ is the hidden layer dimension of the network and $n$ is the number of nodes in the graph G.

***Self-attention layer***: The self-attention layer calculates each node $v_j$ of $g$ based on the single-layer self-attention mechanism of GCN. The attention score of $j \in 1, .., n$ during message propagation, calculated as:

$$s_{g_j} = \sigma(\hat{L}h_j\Theta_{att}), S_g = s_{g_j}{}_{j=1}^{n} \tag{4}$$

$S_g \in \mathbb{R}^{n \times 1}$ is the final attention score matrix of nodes in $g$, where $\hat{L} = \widetilde{D}^{-\frac{1}{2}}\widetilde{A}\widetilde{D}^{-\frac{1}{2}}$, is the Laplacian matrix obtained by normalizing a self connected adjacency matrix $\widetilde{A} = A + I$, $\widetilde{D} \in \mathbb{R}^{n \times n}$ is the degree matrix of $\widetilde{A}$, $\theta_{att} \in \mathbb{R}^{d \times 1}$. In this step, the layer will be based on $S_g$ preserves $\varphi n$ nodes with high weights and masks the hidden states corresponding to the remaining nodes, represented as:

$$idx = Top(S_g, \varphi), H_{V'} = mask(idx, H_V) \tag{5}$$

Where $V' = \{v'_p\}_{p=1}^{\gamma n} \in V$ is the set of nodes with high weights retained in this layer, and $\gamma$ is the retain ratio.

***Graph pooling layer***: Then, the graph pooling layer aggregates node features to form a fixed size representation, and the pooling result is based on the features and topology of the graph. SAGPool uses average maximum pooling to obtain the global graph representation vector $r_g$ of $g$,

$$r_g = Avg(\{h_{v'_p}\}_{p=1}^{\varphi n}) \parallel Max(\{h_{v'_p}\}_{p=1}^{\varphi n}) \tag{6}$$

Where $h_{v'_p} \in H_{V'}$, $\|$ is concatenation.

***Classification layer***: Finally, for the label prediction result $\hat{y}_i$ for $g$ is obtained from the fully connected layer and Softmax, and the expression is as follows:

$$\hat{y}_i = Softmax\left(MLP(r_g)\right) \tag{7}$$

where $\hat{y}_i \in Y$.

## C  Experiment Settings

### C.1  Data Collection

To ensure that the model learns the potential vulnerability patterns implied in the target task as comprehensively as possible and achieves good generalization performance on the source task, the scale of the source task dataset must be larger than that of the sub-task. This aligns with the current research and application scenarios.

All real data in our datasets are sourced from the National Vulnerability Database (NVD), and we collect information about target projects or vulnerability types based on publicly available security commit information related to vulnerabilities. Positive and negative examples are obtained based on the location of the patch before and after program fixes. Moreover, we extract statement-level labels based on the *diff* information of the patches, identifying changes between new and old versions at the statement level within functions.

During data processing, each CPG generated by Joern corresponds to a function in the source file, and each graph association file associated with a source file enables mapping from node IDs in the CPG to source code statements. The experiment employs these preprocessed graph data directly as objects of study. Statistics on the source and target task datasets are provided in Table 6.

Table 6: Dataset statistics

| Dataset | Code Files | Vul : Non-Vul | Label Granularity |
|---------|-----------|---------------|-------------------|
| $S_{119}$ | 6420 | 1:1.1 | Function |
| $S_{416}$ | 3631 | 1:1.5 | Function |
| $T_m$ | 1122 | 1:2.5 | Function, statement |
| $T_{sub}$ | 460 | 1:1.1 | Function, statement |

### C.2  Evaluation Metrics

The evaluation metrics used in the experiment can be calculated as follows:

$$P = \frac{TP}{TP + FP}, \tag{8}$$

$$R = \frac{TP}{TP + FN}, \tag{9}$$

$$F1 = \frac{2 \times P \times R}{P + R}. \tag{10}$$

Here, TP refers to True Positives, TN refers to True Negatives, FP refers to False Positives, and FN refers to False Negatives. Precision measures the accuracy of the model in predicting positive instances, while Recall measures the coverage of the model among all actual positive instances. F1-Score is the harmonic mean of Precision and Recall.

For the evaluation metrics at the statement level, MAP can be calculated as follows:

$$AP_q = \frac{1}{N} \sum_{k=1}^{N} P(K) \times rel_K \tag{11}$$

$$MAP = \frac{1}{Q} \sum_{q=1}^{Q} AP_q \tag{12}$$

Where, $K$ represents the top $K$ statements with the highest vulnerability prediction probabilities, $rel_K$ denotes the number of actual vulnerable statements among the top $K$ statements. $N$ is the total number of samples, and $Q$ is the number of classification categories.

## C.3 Parameter Settings

Table 7: Parameter settings

| Model | Parameter | Setting |
|---|---|---|
| Word2Vec | Min count | 0.001 |
| | Size | 30 |
| | Window | 5 |
| GNNs | Embedding dim | 300 |
| | Hidden dim | 32 |
| | Activation funcion | Relu |
| | Learning rate | 0.0001 |
| | Optimizer | Adam |
| | Train:Validation:Test | 8:1:1 |

During the feature extraction phase, CPGs corresponding to source code files were generated using Joern version 1.1.1033. We employed a pre-trained Word2Vec model for mapping the semantic information of CPG nodes ($V_{semantic}$) to features. The SAGPool model deployed in both source and target tasks were implemented using PyTorch version 1.4.0 and CUDA version 10.2. We conducted all experiments on a workstation equipped with a Quadro RTX 6000 GPU. The model parameters for KF-GVD are detailed in Table 7.

# D Supplementary Experiments

## D.1 Vulnerability Knowledge Sensitivity Analysis

To investigate how the knowledge fusion approach of KF-GVD influences the VD performance on both source and target tasks, we analyze it from the following four aspects:

- *The ratio of samples implement knowledge fusion.*

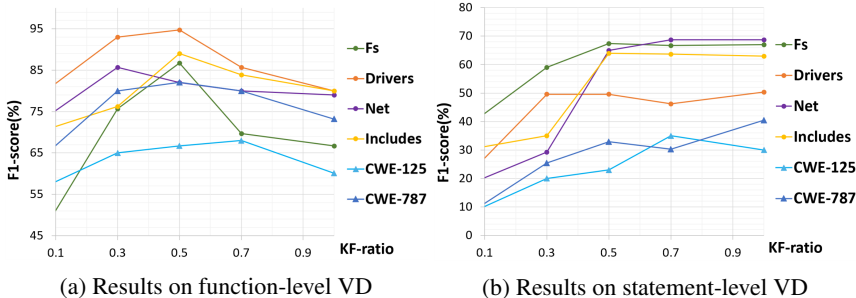

(a) Results on function-level VD      (b) Results on statement-level VD

Figure 7: The F1-score of KF-GVD with knowledge fusion for different ratio of samples on $T_{m_{119}}$ and $T_{sub}$.

We investigate how varying the ratio of fused samples in a specific task affects the overall performance of KF-GVD. Figure 7 shows the influence of the ratio of samples employed knowledge fusion (KF-ratio) on the F1-score for function-level and statement-level VD on the target tasks $T_{m_{119}}$ and $T_{sub}$. It can be observed that, for function-level VD, an increase in the number of fused samples may influence the model's performance to some extent. Similarly, at the statement level, the performance tends to reach a "saturation" state with an increase in fused samples. The model achieves optimal results for both function-level and statement-level detection when the ratio of samples subjected to the knowledge fusion operation is approximately 0.3-0.5 of the total task dataset. According to our statistics, the ratio of samples subjected to fusion in our practice ranges from 0.28-0.56, which aligns with the experimental expectations.

• *Knowledge subgraph fusion coefficient.*

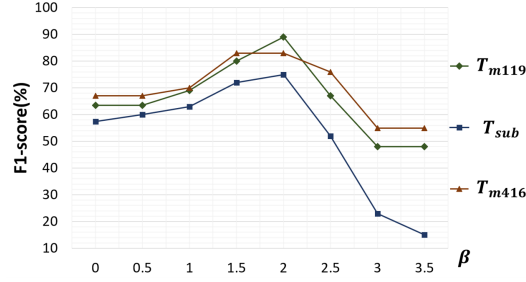

Figure 8: The average F1-score of KF-GVD on different fusion coefficient settings.

In the experiments, we kept the feature values of the nodes $\alpha = 1$ (Equation 1) fixed and varied the fusion coefficient $\beta$ to investigate the impact of vulnerability knowledge on the model's feature learning for specific tasks. To ensure a model configuration with a certain level of consistency, we calculated the average F1-score of KF-GVD for all subtasks within each target task under different fusion coefficient settings. As shown in Figure 8. It can be observed that when the value of $\beta$ is small, it has almost no impact on the overall performance of the model. The model performs well for specific tasks when beta is in the range of 1.5-2. However with the increase of $\beta$ value the influence of relevant knowledge on model feature learning is significant and the performance of the model decreases obviously and is almost ineffective. This also to some extent reflects that the manually introduced vulnerability knowledge adopted by KF-GVD, serving as a supplement to the current task conditions, cannot replace the inherent information in the source code for expressing vulnerability patterns.

• *Performance of the model fused with target task-related knowledge adopted on the source tasks.*

Table 8: Results of applying the target task model to their corresponding source tasks.

(a) The results of applying the models trained on $T_{m_{119}}$ and $T_{sub}$ to $S_{119}$.

(b) The results of applying the models trained on $T_{m_{416}}$ to $S_{416}$.

| Task | P(%) | R(%) | F(%) |
|---|---|---|---|
| $S_{119}$ | 82.9 | 90.9 | 86.7 |
| Fs | 91.7 | 91.1 | 91.4 |
| Drivers | 90.2 | 90.4 | 91.4 |
| Net | 89.6 | 84.9 | 87.2 |
| Includes | 93.7 | 86.4 | 89.9 |
| CWE-125 | 77.1 | 89.8 | 82.9 |
| CWE-787 | 83.2 | 85.0 | 84.1 |
| AD | 5.48 | 2.67 | 0.82 |

| Task | P(%) | R(%) | F(%) |
|---|---|---|---|
| $S_{416}$ | 86.8 | 89.3 | 88.0 |
| Fs | 91.9 | 85.0 | 88.3 |
| Drivers | 89.7 | 88.4 | 89.1 |
| Net | 81.82 | 94.7 | 87.8 |
| Includes | 79.3 | 98.8 | 88.0 |
| Kernel | 95.4 | 83.8 | 89.2 |
| Block | 86.6 | 82.4 | 84.4 |
| AD | -0.76 | -0.5 | -0.17 |

Table 8 shows the results of applying the models trained with fused knowledge from target tasks to their corresponding source tasks. In the last row, we calculate the Average Difference (AD) of evaluation metrics for the models before(in the first row) and after knowledge fusion for the corresponding source tasks. It can be observed that even though KF-GVD incorporates new vulnerability knowledge specific to certain tasks, it still manages to maintain the initial VD performance to some extent on the source tasks, with an improvement on $S_{119}$ and only marginal decreases within 1% for all metrics on $S_{416}$.

• *The impact of implementing different task-oriented vulnerability knowledge.*

We conduct the ablation study on the impact of implementing different task-oriented vulnerability knowledge on the performance of KF-GVD. The knowledge related to specific vulnerability type (vulnerable program operations and sensitive functions) is denoted as K1, and the customized knowledge for specific tasks as K2. Table 9 shows F1-score (%) comparison of VD through the fusion of different vulnerability knowledge on the target tasks corresponding to $S_{119}$. For the target task $T_{m_{119}}$, using only K1 resulted in an average increase of 11.9% in the F1-score, and an average

improvement of 16.3% by K2 only; for the target task $T_{sub}$, K1 resulted in an average increase of 15.1% in the F1-score, and K2 resulted in an average improvement of 7.5%. It can be observed that target tasks like $T_{sub}$ are more sensitive to K1. Furthermore, the combined integration of K1 and K2 resulted in average F1 gains of 23.3% for $T_{m_{119}}$ and 15.7% for $T_{sub}$, further demonstrating the effectiveness of the proposed method.

Table 9: F1-score (%) comparison of VD through the fusion of different vulnerability knowledge on the target tasks corresponding to $S_{119}$.

| Strategy | $T_{m_{119}}$ | | | | $T_{sub}$ | |
|---|---|---|---|---|---|---|
| | Fs | Drivers | Net | Include | CWE-125 | CWE-787 |
| GVD | 65.2 | 76.2 | 64.7 | 59.3 | 54.6 | 64.0 |
| GVD + K1 | 80.4 | 79.1 | 82.9 | 70.6 | 71.3 | 77.5 |
| GVD + K2 | 88.6 | 85.9 | 78.6 | 77.4 | 62.6 | 71.0 |
| GVD + K1 + K2 | 95.7 | 92.3 | 82.5 | 88.0 | 67.9 | 82.1 |

## D.2 Feature Representation Sensitivity Analysis

We evaluate the impact of the feature representation method employed by KF-GVD on VD performance by using different graph representation methods and embedding models on $S_{119}$ and its corresponding target tasks.

● *Graph representation based on different code properties.*

Table 10: F1-score (%) comparison of VD using different graph representations on $S_{119}$ and its corresponding target tasks.

| Graph | $S_{119}$ | $T_{m_{119}}$ | | | | $T_{sub}$ | |
|---|---|---|---|---|---|---|---|
| | | Fs | Drivers | Net | Include | CWE-125 | CWE-787 |
| CFG | 54.4 | 46.8 | 50.8 | 45.2 | 49.3 | 54.6 | 50.3 |
| PDG | 61.8 | 60.7 | 57.9 | 52.3 | 56.8 | 59.6 | 56.5 |
| CPG | 86.7 | 95.7 | 92.3 | 82.5 | 88.0 | 67.9 | 82.1 |

As shown in Table 10, compared to a single code property, the CPG-based code representation method achieves an average F1-score improvement of 27.1% on target tasks. To enable the pre-trained general model to more flexibly adapt to a diverse range of downstream target tasks for enhanced VD, it is crucial to consider more comprehensive vulnerability features during the modeling stage. Although the CPG-based approach may result in slower performance compared to using a single code property during model training, we believe that this trade-off is justified and acceptable, given the potential gains in detection accuracy and model adaptability.

● *Generate node semantic features $V_{semantic}$ using different embedding models.*

Table 11: F1-score (%) comparison of VD using different embedding models on $S_{119}$ and its corresponding target tasks.

| Model | $S_{119}$ | $T_{m_{119}}$ | | | | $T_{sub}$ | |
|---|---|---|---|---|---|---|---|
| | | Fs | Drivers | Net | Include | CWE-125 | CWE-787 |
| Code2Vec | 84.2 | 92.8 | 94.6 | 83.8 | 88.9 | 68.3 | 79.6 |
| Code2Seq | 67.5 | 70.9 | 68.4 | 63.3 | 66.1 | 52.8 | 61.4 |
| Word2Vec | 86.7 | 95.7 | 92.3 | 82.5 | 88.0 | 67.9 | 82.1 |

Table 11 shows the F1-score (%) comparison of VD using different embedding models on $S_{119}$ and its corresponding target tasks. It can be observed that the impact of using Code2Vec and Word2Vec embedding methods on the performance of KF-GVD is very close, and both are superior to the Code2Seq embedding model. However, KF-GVD adopts Word2Vec as the embedding model for generating node semantic feature vectors for the following reasons: First, since the CPG already

incorporates AST properties, we consider the AST parsing process in Code2Vec to be redundant and more time-consuming, whereas Word2Vec is simpler and more efficient in comparison. Second, while Code2Seq leverages attention mechanisms that provide advantages in handling complex code structures and long-distance dependencies. For KF-GVD, the complex structure and dependencies of the source code are represented by CPG as a whole, and the source code corresponding to the CPG node is a concise code snippet or statement generated by joern parsing. We infer that this is also the reason for the relatively poor performance of the Code2Seq model.

### D.3 The Performance of KF-GVD on Cross-Domain Tasks

Table 12: F1-score (%) comparison of VD on cross-domain tasks.

| Method | L->F | L->O | F->L | F->O | O->L | O->F |
|---|---|---|---|---|---|---|
| Cppcheck | 37.5 | 28.2 | 40.1 | 27.6 | 32.4 | 19.0 |
| Flawfinder | 27.5 | 18.6 | 22.8 | 24.1 | 14.8 | 25.9 |
| Sysver | 39.9 | 29.4 | 30.1 | 34.2 | 32.4 | 33.5 |
| VulCNN | 48.4 | 32.3 | 42.3 | 36.4 | 35.9 | 27.3 |
| Devign | 44.6 | 35.1 | 41.2 | 43.1 | 34.3 | 47.5 |
| ReGVD | 54.3 | 33.9 | 38.6 | 40.1 | 44.8 | 50.2 |
| IVDetect | 50.1 | 47.3 | 48.5 | _49.7_ | 42.6 | 48.6 |
| GVD-ft | _55.8_ | _50.0_ | _53.5_ | 48.0 | _62.1_ | _54.7_ |
| **KF-GVD** | 57.1 | 54.0 | 62.4 | 56.8 | 65.6 | 60.3 |

We further verify the generalization of KF-GVD in cross-domain VD scenarios, and conduct cross-domain VD among Linux kernel (L), FFmpeg (F) and Openssl (O). Table 12 shows the comparison of F1 scores (%) on six cross-domain tasks. It can be observed that compared to suboptimal methods, KF-GVD achieves an improvement of 1.3%-8.9% in F1-score, with an average gain of 5.4%. The above experiments demonstrate the effectiveness of KF-GVD in cross-domain VD tasks compared to SOTAs. On the other hand, compared with specific CWE-oriented VD tasks (an average gain of 22.6% in F1-score), KF-GVD performs relatively poorly in cross-domain tasks. The unknown vulnerability patterns between cross-domain projects, and the significant differences in programming paradigms and code functionality across projects pose challenges to effective knowledge transfer.

